# Mixing Properties of Conditional Markov Chains with Unbounded Feature Functions

**Mathieu Sinn**
IBM Research - Ireland
Mulhuddart, Dublin 15
mathsinn@ie.ibm.com

**Bei Chen**
McMaster University
Hamilton, Ontario, Canada
bei.chen@math.mcmaster.ca

## Abstract

Conditional Markov Chains (also known as Linear-Chain Conditional Random Fields in the literature) are a versatile class of discriminative models for the distribution of a sequence of hidden states conditional on a sequence of observable variables. Large-sample properties of Conditional Markov Chains have been first studied in [1]. The paper extends this work in two directions: first, mixing properties of models with unbounded feature functions are being established; second, necessary conditions for model identifiability and the uniqueness of maximum likelihood estimates are being given.

## 1 Introduction

Conditional Random Fields (CRF) are a widely popular class of discriminative models for the distribution of a set of hidden states conditional on a set of observable variables. The fundamental assumption is that the hidden states, conditional on the observations, form a Markov random field [2,3]. Of special importance, particularly for the modeling of sequential data, is the case where the underlying undirected graphical model forms a simple linear chain. In the literature, this subclass of models is often referred to as Linear-Chain Conditional Random Fields. This paper adopts the terminology of [4] and refers to such models as Conditional Markov Chains (CMC).

Large-sample properties of CRFs and CMCs have been first studied in [1] and [5]. [1] defines CMCs of infinite length and studies ergodic properties of the joint sequences of observations and hidden states. The analysis relies on fundamental results from the theory of weak ergodicity [6]. The exposition is restricted to CMCs with bounded feature functions which precludes the application, e.g., to models with linear features and Gaussian observations. [5] considers weak consistency and central limit theorems for models with a more general structure. Ergodicity and mixing of the models is assumed, but no explicit conditions on the feature functions or on the distribution of the observations are given. An analysis of model identifiability in the case of finite sequences can be found in [7].

The present paper studies mixing properties of Conditional Markov Chains with unbounded feature functions. The results are fundamental for analyzing the consistency of Maximum Likelihood estimates and establishing Central Limit Theorems (which are very useful for constructing statistical hypothesis tests, e.g., for model misspecificiations and the signficance of features). The paper is organized as follows: Sec. 2 reviews the definition of infinite CMCs and some of their basic properties. In Sec. 3 the ergodicity results from [1] are extended to models with unbounded feature functions. Sec. 4 establishes various mixing properties. A key result is that, in order to allow for unbounded feature functions, the observations need to follow a distribution such that Hoeffding-type concentration inequalities can be established. Furthermore, the mixing rates depend on the tail behaviour of the distribution. In Sec. 5 the mixture properties are used to analyze model identifiability and consistency of the Maximum Likelihood estimates. Sec. 6 concludes with an outlook on open problems for future research.

## 2 Conditional Markov Chains

**Preliminaries.** We use $\mathbb{N}$, $\mathbb{Z}$ and $\mathbb{R}$ to denote the sets of natural numbers, integers and real numbers, respectively. Let $\mathcal{X}$ be a metric space with the Borel sigma-field $\mathcal{A}$, and $\mathcal{Y}$ be a finite set. Furthermore, consider a probability space $(\Omega, \mathcal{F}, \mathbb{P})$ and let $\boldsymbol{X} = (X_t)_{t \in \mathbb{Z}}$, $\boldsymbol{Y} = (Y_t)_{t \in \mathbb{Z}}$ be sequences of measurable mappings from $\Omega$ into $\mathcal{X}$ and $\mathcal{Y}$, respectively. Here,

- $\boldsymbol{X}$ is an infinite sequence of *observations* ranging in the domain $\mathcal{X}$,

- $\boldsymbol{Y}$ is an aligned sequence of *hidden states* taking values in the finite set $\mathcal{Y}$.

For now, the distribution of $\boldsymbol{X}$ is arbitrary. Next we define Conditional Markov Chains, which parameterize the conditional distribution of $\boldsymbol{Y}$ given $\boldsymbol{X}$.

**Definition.** Consider a vector $\boldsymbol{f}$ of real-valued functions $f : \mathcal{X} \times \mathcal{Y} \times \mathcal{Y} \to \mathbb{R}$, called the *feature functions*. Throughout this paper, we assume that the following condition is satisfied:

(A1) All feature functions are finite: $|f(x, i, j)| < \infty$ for all $x \in \mathcal{X}$ and $i, j \in \mathcal{Y}$.

Associated with the feature functions is a vector $\boldsymbol{\lambda}$ of real-valued *model-weights*. The key in the definition of Conditional Markov Chains is the matrix $\boldsymbol{M}(x)$ with the $(i, j)$-th component

$$m(x, i, j) \;=\; \exp(\boldsymbol{\lambda}^T \boldsymbol{f}(x, i, j)).$$

In terms of statistical physics, $m(x, i, j)$ measures the *potential* of the transition between the hidden states $i$ and $j$ from time $t-1$ to $t$, given the observation $x$ at time $t$. Next, for a sequence $\boldsymbol{x} = (x_t)_{t \in \mathbb{Z}}$ in $\mathcal{X}$ and time points $s, t \in \mathbb{Z}$ with $s \leq t$, introduce the vectors

$$\begin{aligned}
\boldsymbol{\alpha}_s^t(\boldsymbol{x}) &= \boldsymbol{M}(x_t)^T \dots \boldsymbol{M}(x_s)^T (1, 1, \dots, 1)^T, \\
\boldsymbol{\beta}_s^t(\boldsymbol{x}) &= \boldsymbol{M}(x_{s+1}) \dots \boldsymbol{M}(x_t) (1, 1, \dots, 1)^T,
\end{aligned}$$

and write $\alpha_s^t(\boldsymbol{x}, i)$ and $\beta_s^t(\boldsymbol{x}, j)$ to denote the $i$th respectively $j$th components. Intuitively, $\alpha_s^t(\boldsymbol{x}, i)$ measures the potential of the hidden state $i$ at time $t$ given the observations $x_s, \dots, x_t$ and assuming that at time $s - 1$ all hidden states have potential equal to 1. Similarly, $\beta_s^t(\boldsymbol{x}, j)$ is the potential of $j$ at time $s$ assuming equal potential of all hidden states at time $t$. Now let $t \in \mathbb{Z}$ and $k \in \mathbb{N}$, and define the distribution of the labels $Y_t, \dots, Y_{t+k}$ conditional on $\boldsymbol{X}$,

$$\begin{aligned}
\mathbb{P}(Y_t = y_t, \dots, Y_{t+k} = y_{t+k} \mid \boldsymbol{X}) \;&:=\; \prod_{i=1}^{k} m(X_{t+i}, y_{t+i-1}, y_{t+i}) \\
&\times \lim_{n \to \infty} \frac{\alpha_{t-n}^t(\boldsymbol{X}, y_t)\, \beta_{t+k}^{t+k+n}(\boldsymbol{X}, y_{t+k})}{\boldsymbol{\alpha}_{t-n}^t(\boldsymbol{X})^T \boldsymbol{\beta}_t^{t+k+n}(\boldsymbol{X})}.
\end{aligned}$$

Note that, under assumption (A1), the limit on the right hand side is well-defined (see Theorem 2 in [1]). Furthermore, the family of all marginal distributions obtained this way satisfies the consistency conditions of Kolmogorov's Extension Theorem. Hence we obtain a unique distribution for $\boldsymbol{Y}$ conditional on $\boldsymbol{X}$ parameterized by the feature functions $\boldsymbol{f}$ and the model weights $\boldsymbol{\lambda}$. Intuitively, the distribution is obtained by conditioning the marginal distributions of $\boldsymbol{Y}$ on the finite observational context $(X_{t-n}, \dots, X_{t+k+n})$, and then letting the size of the context going to infinity.

**Basic properties.** We introduce the following notation: For any matrix $\boldsymbol{P} = (p_{ij})$ with strictly positive entries let $\phi(\boldsymbol{P})$ denote the mixing coefficient

$$\phi(\boldsymbol{P}) \;=\; \min_{i,j,k,l} \frac{p_{ik} p_{jl}}{p_{jk} p_{il}}.$$

Note that $0 \leq \phi(\boldsymbol{P}) \leq 1$. This coefficient will play a key role in the analysis of mixing properties. The following proposition summarizes fundamental properties of the distribution of $\boldsymbol{Y}$ conditional on $\boldsymbol{X}$, which directly follow from the above definition (also see Corollary 1 in [1]).

**Proposition 1.** *Suppose that condition* (A1) *holds true. Then* $\boldsymbol{Y}$ *conditional on* $\boldsymbol{X}$ *forms a time-inhomogeneous Markov chain. Moreover, if* $\boldsymbol{X}$ *is strictly stationary, then the joint distribution of the aligned sequences* $(\boldsymbol{X}, \boldsymbol{Y})$ *is strictly stationary. The conditional transition probabilities* $P_t(\boldsymbol{x}, i, j) := \mathbb{P}(Y_t = j \mid Y_{t-1} = i, \boldsymbol{X} = \boldsymbol{x})$ *of* $\boldsymbol{Y}$ *given* $\boldsymbol{X} = \boldsymbol{x}$ *have the following form:*

$$P_t(\boldsymbol{x}, i, j) = m(x_t, i, j) \lim_{n \to \infty} \frac{\beta_t^n(\boldsymbol{x}, j)}{\beta_{t-1}^n(\boldsymbol{x}, i)}.$$

*In particular, a lower bound for* $P_t(\boldsymbol{x}, i, j)$ *is given by*

$$P_t(\boldsymbol{x}, i, j) \geq \frac{m(x_t, i, j) \left( \min_{k \in \mathcal{Y}} m(x_{t+1}, i, k) \right)}{|\mathcal{Y}| \left( \max_{k \in \mathcal{Y}} m(x_t, j, k) \right) \left( \max_{k,l \in \mathcal{Y}} m(x_{t+1}, k, l) \right)},$$

*and the matrix of transition probabilities* $\boldsymbol{P}_t(\boldsymbol{x})$*, with the* $(i, j)$*-th component given by* $P_t(\boldsymbol{x}, i, j)$*, satisfies* $\phi(\boldsymbol{P}_t(\boldsymbol{x})) = \phi(\boldsymbol{M}(x_t))$*.*

## 3 Ergodicity

In this section we establish conditions under which the aligned sequences $(\boldsymbol{X}, \boldsymbol{Y})$ are jointly ergodic. Let us first recall the definition of ergodicity of $\boldsymbol{X}$ (see [8]): By $\mathcal{X}$ we denote the space of sequences $\boldsymbol{x} = (x_t)_{t \in \mathbb{Z}}$ in $\mathcal{X}$, and by $\mathcal{A}$ the corresponding product $\sigma$-field. Consider the probability measure $P_{\boldsymbol{X}}$ on $(\mathcal{X}, \mathcal{A})$ defined by $P_{\boldsymbol{X}}(\boldsymbol{A}) := \mathbb{P}(\boldsymbol{X} \in \boldsymbol{A})$ for $\boldsymbol{A} \in \mathcal{A}$. Finally, let $\tau$ denote the operator on $\mathcal{X}$ which shifts sequences one position to the left: $\tau \boldsymbol{x} = (x_{t+1})_{t \in \mathbb{Z}}$. Then ergodicity of $\boldsymbol{X}$ is formally defined as follows:

(A2) $\boldsymbol{X}$ is ergodic, that is, $P_{\boldsymbol{X}}(\boldsymbol{A}) = P_{\boldsymbol{X}}(\tau^{-1} \boldsymbol{A})$ for every $\boldsymbol{A} \in \mathcal{A}$, and $P_{\boldsymbol{X}}(\boldsymbol{A}) \in \{0, 1\}$ for every set $\boldsymbol{A} \in \mathcal{A}$ satisfying $\boldsymbol{A} = \tau^{-1} \boldsymbol{A}$.

As a particular consequence of the invariance $P_{\boldsymbol{X}}(\boldsymbol{A}) = P_{\boldsymbol{X}}(\tau^{-1} \boldsymbol{A})$, we obtain that $\boldsymbol{X}$ is strictly stationary. Now we are able to formulate the key result of this section, which will be of central importance in the later analysis. For simplicity, we state it for functions depending on the values of $\boldsymbol{X}$ and $\boldsymbol{Y}$ only at time $t$. The generalization of the statement is straight-forward. In our later analysis, we will use the theorem to show that the time average of feature functions $f(X_t, Y_{t-1}, Y_t)$ converges to the expected value $\mathbb{E}[f(X_t, Y_{t-1}, Y_t)]$.

**Theorem 1.** *Suppose that conditions* (A1) *and* (A2) *hold, and* $g : \mathcal{X} \times \mathcal{Y} \to \mathbb{R}$ *is a function which satisfies* $\mathbb{E}[|g(X_t, Y_t)|] < \infty$*. Then*

$$\lim_{n \to \infty} \frac{1}{n} \sum_{t=1}^{n} g(X_t, Y_t) = \mathbb{E}[g(X_t, Y_t)] \quad \mathbb{P}\text{-almost surely.}$$

*Proof.* Consider the sequence $\boldsymbol{Z} = (Z_t)_{t \in \mathbb{N}}$ given by $Z_t := (\tau^{t-1} \boldsymbol{X}, Y_t)$, where we write $\tau^{t-1}$ to denote the $(t-1)$th iterate of $\tau$. Note that $Z_t$ represents the hidden state at time $t$ together with the entire aligned sequence of observations $\tau^{t-1} \boldsymbol{X}$. In the literature, such models are known as Markov sequences in random environments (see [9]). The key step in the proof is to show that $\boldsymbol{Z}$ is ergodic. Then, for any function $h : \mathcal{X} \times \mathcal{Y} \to \mathbb{R}$ with $\mathbb{E}[|h(Z_t)|] < \infty$, the time average $\frac{1}{n} \sum_{t=1}^{n} h(Z_t)$ converges to the expected value $\mathbb{E}[h(Z_t)]$ $\mathbb{P}$-almost surely. Applying this result to the composition of the function $g$ and the projection of $(\tau^{t-1} \boldsymbol{X}, Y_t)$ onto $(X_t, Y_t)$ completes the proof. The details of the proof that $\boldsymbol{Z}$ is ergodic can be found in an extended version of this paper, which is included in the supplementary material. □

## 4 Mixing properties

In this section we are going to study mixing properties of the aligned sequences $(\boldsymbol{X}, \boldsymbol{Y})$. To establish the results, we will assume that the distribution of the observations $\boldsymbol{X}$ satisfies conditions under which certain concentration inequalities hold true:

(A3) Let $A \subset \mathcal{A}$ be a measurable set, with $p := \mathbb{P}(X_t \in A)$ and $S_n(\boldsymbol{x}) := \frac{1}{n} \sum_{t=1}^{n} \mathbf{1}(x_t \in A)$ for $\boldsymbol{x} \in \mathcal{X}$. Then there exists a constant $\gamma$ such that, for all $n \in \mathbb{N}$ and $\epsilon > 0$,

$$\mathbb{P}(|S_n(\boldsymbol{X}) - p| \geq \epsilon) \leq \exp(-\gamma \epsilon^2 n).$$

If $\boldsymbol{X}$ is a sequence of independent random variables, then (A3) follows by Hoeffding's inequality. In the dependent case, concentration inequalities of this type can be obtained by imposing Martingale or mixing conditions on $\boldsymbol{X}$ (see [12,13]). Furthermore, we will make the following assumption, which relates the feature functions to the tail behaviour of the distribution of $\boldsymbol{X}$:

(A4) Let $h : [0, \infty) \to [0, \infty)$ be a differentiable decreasing function with $h(z) = O(z^{-(1+\kappa)})$ for some $\kappa > 0$. Furthermore, let

$$F(x) \quad := \quad \sum_{j,k \in \mathcal{Y}} |\boldsymbol{\lambda}^T \boldsymbol{f}(x, j, k)|$$

for $x \in \mathcal{X}$. Then $\mathbb{E}[h(F(X_t))^{-1}]$ and $\mathbb{E}[h'(F(X_t))^{-1}]$ both exist and are finite.

The following theorem establishes conditions under which the expected conditional covariances of square-integrable functions are summable. The result is obtained by studying ergodic properties of the transition probability matrices.

**Theorem 2.** *Suppose that conditions* (A1) - (A3) *hold true, and* $g : \mathcal{X} \times \mathcal{Y} \to \mathbb{R}$ *is a function with finite second moment,* $\mathbb{E}[|g(X_t, Y_t)|^2] < \infty$. *Let* $\gamma_{t,k}(\boldsymbol{X}) = \mathrm{Cov}(g(X_t, Y_t), g(X_{t+k}, Y_{t+k}) \,|\, \boldsymbol{X})$ *denote the covariance of* $g(X_t, Y_t)$ *and* $g(X_{t+k}, Y_{t+k})$ *conditional on* $\boldsymbol{X}$. *Then, for every* $t \in \mathbb{Z}$:

$$\lim_{n \to \infty} \sum_{k=1}^{n} \mathbb{E}[|\gamma_{t,k}(\boldsymbol{X})|] \quad < \quad \infty.$$

*Proof.* Without loss of generality we may assume that $g$ can be written as $g(x, y) = g(x)\boldsymbol{1}(y = i)$. Hence, using Hölder's inequality, we obtain

$$\mathbb{E}[|\gamma_{t,k}(\boldsymbol{X})|] \quad \leq \quad \mathbb{E}[|g(X_t)|]\, \mathbb{E}[|g(X_{t+k})|]\, \mathbb{E}[|\mathrm{Cov}(\boldsymbol{1}(Y_t = i), \boldsymbol{1}(Y_{t+k} = i) \,|\, \boldsymbol{X})|].$$

According to the assumptions, we have $\mathbb{E}[|g(X_t)|] = \mathbb{E}[|g(X_{t+k})|] < \infty$, so we only need to bound the expectation of the conditional covariance. Note that

$$\mathrm{Cov}(\boldsymbol{1}(Y_t = i), \boldsymbol{1}(Y_{t+k} = i) \,|\, \boldsymbol{X}) \;=\; \mathbb{P}(Y_t = i, Y_{t+k} = i \,|\, \boldsymbol{X}) - \mathbb{P}(Y_t = i \,|\, \boldsymbol{X})\, \mathbb{P}(Y_{t+k} = i \,|\, \boldsymbol{X}).$$

Recall the definition of $\phi(\boldsymbol{P})$ before Corollary 1. Using probabilistic arguments, it is not difficult to show that the ratio of $\mathbb{P}(Y_t = i, Y_{t+k} = i \,|\, \boldsymbol{X})$ to $\mathbb{P}(Y_t = i \,|\, \boldsymbol{X})\, \mathbb{P}(Y_{t+k} = i \,|\, \boldsymbol{X})$ is greater than or equal to $\phi(\boldsymbol{P}_{t+1}(\boldsymbol{X}) \ldots \boldsymbol{P}_{t+k}(\boldsymbol{X}))$, where $\boldsymbol{P}_{t+1}(\boldsymbol{X}), \ldots, \boldsymbol{P}_{t+k}(\boldsymbol{X})$ denote the transition matrices introduced in Proposition 1. Hence,

$$|\mathrm{Cov}(\boldsymbol{1}(Y_t = i), \boldsymbol{1}(Y_{t+k} = i) \,|\, \boldsymbol{X})| \;\leq\; \mathbb{P}(Y_t = i, Y_{t+k} = i \,|\, \boldsymbol{X})[1 - \phi(\boldsymbol{P}_{t+1}(\boldsymbol{X}) \ldots \boldsymbol{P}_{t+k}(\boldsymbol{X}))].$$

Now, using results from the theory of weak ergodicity (see Chapter 3 in [6]), we can establish

$$\frac{1 - \sqrt{\phi(\boldsymbol{P}_{t+1}(\boldsymbol{x}) \ldots \boldsymbol{P}_{t+k}(\boldsymbol{x}))}}{1 + \sqrt{\phi(\boldsymbol{P}_{t+1}(\boldsymbol{x}) \ldots \boldsymbol{P}_{t+k}(\boldsymbol{x}))}} \quad \leq \quad \prod_{j=1}^{k} \frac{1 - \sqrt{\phi(\boldsymbol{P}_{t+j}(\boldsymbol{x}))}}{1 + \sqrt{\phi(\boldsymbol{P}_{t+j}(\boldsymbol{x}))}}$$

for all $\boldsymbol{x} \in \mathcal{X}$. Using Bernoulli's inequality and the fact $\phi(\boldsymbol{P}_{t+j}(\boldsymbol{x})) = M(x_{t+j})$ established in Proposition 1, we obtain $\phi(\boldsymbol{P}_{t+1}(\boldsymbol{x}) \ldots \boldsymbol{P}_{t+k}(\boldsymbol{x})) \geq 1 - 4 \prod_{j=1}^{k}[1 - \phi(M(x_{t+j}))]$. Consequently,

$$|\mathrm{Cov}(\boldsymbol{1}(Y_t = i), \boldsymbol{1}(Y_{t+k} = i) \,|\, \boldsymbol{X})| \quad \leq \quad 4 \prod_{j=1}^{k} [1 - \phi(M(X_{t+j}))].$$

With the notation introduced in assumption (A3), let $\delta > 0$ and $A \subset \mathcal{X}$ with $p > 0$ be such that $x \in A$ implies $\phi(M(x)) \geq \delta$. Furthermore, let $\epsilon$ be a constant with $0 < \epsilon < p$. In order to bound $|\mathrm{Cov}(\boldsymbol{1}(Y_t = i), \boldsymbol{1}(Y_{t+k} = i) \,|\, \boldsymbol{X})|$ for a given $k \in \mathbb{N}$, we distinguish two different cases: If $|S_k(\boldsymbol{X}) - p| < \epsilon$, then we obtain

$$4 \prod_{j=1}^{k} \left(1 - \phi(M(X_{t+j}))\right) \quad \leq \quad 4\,(1 - \delta)^{k(p-\epsilon)}.$$

If $|S_k(\boldsymbol{X}) - p| \geq \epsilon$, then we use the trivial upper bound 1. According to assumption (A3), the probability of the latter event is bounded by an exponential, and hence

$$\mathbb{E}[|\mathrm{Cov}(\boldsymbol{1}(Y_t = i), \boldsymbol{1}(Y_{t+k} = i) \,|\, \boldsymbol{X})|] \quad \leq \quad 4\,(1 - \delta)^{k(p-\epsilon)} + \exp(-\gamma\,\epsilon^2 k).$$

Obviously, the sum of all these expectations is finite, which completes the proof. $\qquad\square$

The next theorem bounds the difference between the distribution of $\boldsymbol{Y}$ conditional on $\boldsymbol{X}$ and finite approximations of it. Introduce the following notation: For $t, k \geq 0$ with $t + k \leq n$ let

$$\mathbb{P}^{(0:n)}(Y_t = y_t, \ldots, Y_{t+k} = y_{t+k} \mid \boldsymbol{X} = \boldsymbol{x})$$

$$:= \prod_{i=1}^{k} m(x_{t+i}, y_{t+i-1}, y_{t+i}) \lim_{n \to \infty} \frac{\alpha_0^t(\boldsymbol{x}, y_t) \, \beta_{t+k}^n(\boldsymbol{x}, y_{t+k})}{\boldsymbol{\alpha}_0^t(\boldsymbol{x})^T \boldsymbol{\beta}_t^n(\boldsymbol{x})}.$$

Accordingly, write $\mathbb{E}^{(0:n)}$ for expectations taken with respect to $\mathbb{P}^{(0:n)}$. As emphasized by the superscrips, these quantities can be regarded as marginal distributions of $\boldsymbol{Y}$ conditional on the observations at times $t = 0, 1, \ldots, n$. To simplify notation, the following theorem is stated for 1-dimensional conditional marginal distributions, however, the extension to the general case is straight-forward.

**Theorem 3.** *Suppose that conditions* (A1) - (A4) *hold true. Then the limit*

$$\mathbb{P}^{(0:\infty)}(Y_t = i \mid \boldsymbol{X}) \quad := \quad \lim_{n \to \infty} \mathbb{P}^{(0:n)}(Y_t = i \mid \boldsymbol{X})$$

*is well-defined $\mathbb{P}$-almost surely. Moreover, there exists a measurable function $C(\boldsymbol{x})$ of $\boldsymbol{x} \in \mathcal{X}$ with finite expectation, $E[\|C(\boldsymbol{X})\|] < \infty$, and a function $h(z)$ satisfying the conditions in* (A4) , *such that*

$$\left| \mathbb{P}^{(0:\infty)}(Y_t = i \mid \boldsymbol{X}) - \mathbb{P}^{(0:n)}(Y_t = i \mid \boldsymbol{X}) \right| \quad \leq \quad C(\tau^t \boldsymbol{X}) \, h(n - t).$$

*Proof.* Define $\boldsymbol{G}_n(\boldsymbol{x}) := \boldsymbol{M}(x_{t+1}) \ldots \boldsymbol{M}(x_n)$ and write $g_n(\boldsymbol{x}, i, j)$ for the $(i, j)$-th component of $\boldsymbol{G}_n(\boldsymbol{x})$. Note that $\boldsymbol{\beta}_t^n(\boldsymbol{x}) = \boldsymbol{G}_n(\boldsymbol{x})(1, 1, \ldots, 1)^T$. According to Lemma 3.4 in [6], there exist numbers $r_{ij}(\boldsymbol{x})$ such that

$$\lim_{n \to \infty} \frac{g_n(\boldsymbol{x}, i, k)}{g_n(\boldsymbol{x} j, k)} \quad = \quad r_{ij}(\boldsymbol{x})$$

for all $k \in \mathcal{Y}$. Consequently, the ratio of $\beta_t^n(\boldsymbol{x}, i)$ to $\beta_t^n(\boldsymbol{x}, j)$ converges to $r_{ij}(\boldsymbol{x})$, and hence

$$\lim_{n \to \infty} \frac{\alpha_0^t(\boldsymbol{x}, i) \, \beta_t^n(\boldsymbol{x}, i)}{\boldsymbol{\alpha}_0^t(\boldsymbol{x})^T \boldsymbol{\beta}_t^n(\boldsymbol{x})} \quad = \quad \frac{1}{\boldsymbol{q}_i(\boldsymbol{x})^T \boldsymbol{r}_i(\boldsymbol{x})}$$

where we use the notation $\boldsymbol{q}_i(\boldsymbol{x}) = \boldsymbol{\alpha}_0^t(\boldsymbol{x}) / \alpha_0^t(\boldsymbol{x}, i)$ and $\boldsymbol{r}_i(\boldsymbol{x})$ denotes the vector with the $k$th component $r_{ki}(\boldsymbol{x})$. This proves the first part of the theorem. In order to prove the second part, note that $|x - y| \leq |x^{-1} - y^{-1}|$ for any $x, y \in (0, 1]$, and hence

$$\left| \mathbb{P}^{(0:\infty)}(Y_t = i \mid \boldsymbol{X}) - \mathbb{P}^{(0:n)}(Y_t = i \mid \boldsymbol{X}) \right| \quad \leq \quad \left| \boldsymbol{q}_i(\boldsymbol{X})^T \boldsymbol{r}_i(\boldsymbol{X}) - \frac{\boldsymbol{\alpha}_0^t(\boldsymbol{X})^T \boldsymbol{\beta}_t^n(\boldsymbol{X})}{\alpha_0^t(\boldsymbol{X}, i) \, \beta_t^n(\boldsymbol{X}, i)} \right|.$$

To bound the latter expression, introduce the vectors $\underline{\boldsymbol{r}}_i^n(\boldsymbol{x})$ and $\overline{\boldsymbol{r}}_i^n(\boldsymbol{x})$ with the $k$th components

$$\underline{r}_{ki}^n(\boldsymbol{x}) = \min_{l \in \mathcal{Y}} \left( \frac{g_n(\boldsymbol{x}, k, l)}{g_n(\boldsymbol{x}, i, l)} \right) \quad \text{and} \quad \overline{r}_{ki}^n(\boldsymbol{x}) = \max_{l \in \mathcal{Y}} \left( \frac{g_n(\boldsymbol{x}, k, l)}{g_n(\boldsymbol{x}, i, l)} \right).$$

It is easy to see that $\boldsymbol{q}_i(\boldsymbol{x})^T \underline{\boldsymbol{r}}_i^n(\boldsymbol{x}) \leq \boldsymbol{q}_i(\boldsymbol{x})^T \boldsymbol{r}_i(\boldsymbol{x}) \leq \boldsymbol{q}_i(\boldsymbol{x})^T \overline{\boldsymbol{r}}_i^n(\boldsymbol{x})$, and

$$\boldsymbol{q}_i(\boldsymbol{x})^T \underline{\boldsymbol{r}}_i^n(\boldsymbol{x}) \quad \leq \quad \frac{\boldsymbol{\alpha}_0^t(\boldsymbol{x})^T \boldsymbol{\beta}_t^n(\boldsymbol{x})}{\alpha_0^t(\boldsymbol{x}, i) \, \beta_t^n(\boldsymbol{x}, i)} \quad \leq \quad \boldsymbol{q}_i(\boldsymbol{x})^T \overline{\boldsymbol{r}}_i^n(\boldsymbol{x}).$$

Hence,

$$\left| \boldsymbol{q}_i(\boldsymbol{X})^T \boldsymbol{r}_i(\boldsymbol{X}) - \frac{\boldsymbol{\alpha}_0^t(\boldsymbol{X})^T \boldsymbol{\beta}_t^n(\boldsymbol{X})}{\alpha_0^t(\boldsymbol{X}, i) \, \beta_t^n(\boldsymbol{X}, i)} \right| \quad \leq \quad \left| \boldsymbol{q}_i(\boldsymbol{X})^T \left( \overline{\boldsymbol{r}}_i^n(\boldsymbol{X}) - \underline{\boldsymbol{r}}_i^n(\boldsymbol{X}) \right) \right|.$$

Due to space limitations, we only give a sketch of the proof how the latter quantity can be bounded. For details, see the extended version of this paper in the supplementary material. The first step is to show the existence of a function $C_1(\boldsymbol{x})$ with $\mathbb{E}[\|C_1(\boldsymbol{X})\|] < \infty$ such that $|\underline{r}_{ki}^n(\boldsymbol{X}) - \overline{r}_{ki}^n(\boldsymbol{X})| \leq C_1(\tau^t \boldsymbol{X}) (1 - \zeta)^{n-t}$ for some $\zeta > 0$. With the function $F(x)$ introduced in assumption (A4), we define $C_2(x) := \exp(F(x))$ for $x \in \mathcal{X}$ and arrive at

$$\left| \mathbb{P}^{(0:\infty)}(Y_t = i \mid \boldsymbol{X}) - \mathbb{P}^{(0:n)}(Y_t = i \mid \boldsymbol{X}) \right| \quad \leq \quad |\mathcal{Y}|^2 \, C_1(\tau^t \boldsymbol{X}) \, C_2(X_t) \, (1 - \zeta)^{n-t}.$$

The next step is to construct a function $C_3(x)$ satisfying the following two conditions: $(i)$ If $C_2(x)(1 - \zeta)^k \geq 1$, then $C_3(x)h(k) \geq 1$. $(ii)$ If $C_2(x)(1 - \zeta)^k < 1$, then $C_3(x)h(k) \geq C_2(x)(1 - \zeta)^k$. Since the difference between two probabilities cannot exceed 1, we obtain

$$\left| \mathbb{P}^{(0:\infty)}(Y_t = i \mid \boldsymbol{X}) - \mathbb{P}^{(0:n)}(Y_t = i \mid \boldsymbol{X}) \right| \leq |\mathcal{Y}|^2 \, C_1(\tau^t \boldsymbol{X}) \, C_3(X_t) \, h(n-t).$$

The last step is to show that $\mathbb{E}[|C_3(X_t)|] < \infty$. $\qquad \square$

The following result will play a key role in the later analysis of empirical likelihood functions.

**Theorem 4.** *Suppose that conditions* (A1) - (A4) *hold, and the function* $g : \mathcal{X} \times \mathcal{Y} \to \mathbb{R}$ *satisfies* $\mathbb{E}[|g(X_t, Y_t)|] < \infty$. *Then*

$$\lim_{n \to \infty} \frac{1}{n} \sum_{t=1}^{n} \mathbb{E}^{(0:n)}[g(X_t, Y_t) \mid \boldsymbol{X}] = \mathbb{E}[g(X_t, Y_t)] \quad \mathbb{P}\text{-almost surely.}$$

*Proof.* Without loss of generality we may assume that $g$ can be written as $g(x, y) = g(x)\mathbf{1}(y = i)$. Using the result from Theorem 3, we obtain

$$\left| \sum_{t=1}^{n} \mathbb{E}^{(0:n)}[g(X_t, Y_t) \mid \boldsymbol{X}] - \sum_{t=1}^{n} \mathbb{E}^{(0:\infty)}[g(X_t, Y_t) \mid \boldsymbol{X}] \right| \leq \sum_{t=1}^{n} |g(X_t)| \, |C(\tau^t \boldsymbol{X})| \, h(n-t),$$

where $h(z)$ is a function satisfying the conditions in assumption (A4). See the extended version of this paper in the supplementary material for more details. Using the facts that $\boldsymbol{X}$ is ergodic and the expectations of $|g(X_t)|$ and $|C(\tau^t \boldsymbol{X})|$ are finite, we obtain

$$\lim_{n \to \infty} \frac{1}{n} \left| \sum_{t=1}^{n} \mathbb{E}^{(0:n)}[g(X_t, Y_t) \mid \boldsymbol{X}] - \sum_{t=1}^{n} \mathbb{E}^{(0:\infty)}[g(X_t, Y_t) \mid \boldsymbol{X}] \right| = 0.$$

By similar arguments to the proof of the first part of Theorem 3 one can show that the difference $|\mathbb{E}^{(0:\infty)}[g(X_t, Y_t) \mid \boldsymbol{X}] - \mathbb{E}[g(X_t, Y_t) \mid \boldsymbol{X}]|$ tends to 0 as $t \to \infty$. Thus,

$$\lim_{n \to \infty} \frac{1}{n} \left| \sum_{t=1}^{n} \mathbb{E}^{(0:\infty)}[g(X_t, Y_t) \mid \boldsymbol{X}] - \sum_{t=1}^{n} \mathbb{E}[g(X_t, Y_t) \mid \boldsymbol{X}] \right| = 0.$$

Now, noting that $E[g(X_t, Y_t) \mid \boldsymbol{X}] = E[g(X_0, Y_0) \mid \tau^t \boldsymbol{X}]$ for every $t$, the theorem follows by the ergodicity of $\boldsymbol{X}$. $\qquad \square$

## 5 Maximum Likelihood learning and model identifiability

In this section we apply the previous results to analyze asymptotic properties of the empirical likelihood function. The setting is the following: Suppose that we observe finite subsequences $\boldsymbol{X}_n = (X_0, \ldots, X_n)$ and $\boldsymbol{Y}_n = (Y_0, \ldots, Y_n)$ of $\boldsymbol{X}$ and $\boldsymbol{Y}$, where the distribution of $\boldsymbol{Y}$ conditional on $\boldsymbol{X}$ follows a conditional Markov chain with fixed feature functions $\boldsymbol{f}$ and unknown model weights $\boldsymbol{\lambda}_*$. We assume that $\boldsymbol{\lambda}_*$ lies in some parameter space $\boldsymbol{\Theta}$, the structure of which will become important later. To emphasize the role of the model weights, we will use subscripts, e.g., $\mathbb{P}_{\boldsymbol{\lambda}}$ and $\mathbb{E}_{\boldsymbol{\lambda}}$, to denote the conditional distributions. Our goal is to identify the unknown model weights from the finite samples, $\boldsymbol{X}_n$ and $\boldsymbol{Y}_n$. In order to do so, introduce the shorthand notation $\boldsymbol{f}(\boldsymbol{x}_n, \boldsymbol{y}_n) = \sum_{t=1}^{n} \boldsymbol{f}(x_t, y_{t-1}, y_t)$ for $\boldsymbol{x}_n = (x_0, \ldots, x_n)$ and $\boldsymbol{y}_n = (y_0, \ldots, y_n)$. Consider the normalized conditional likelihood,

$$\mathcal{L}_n(\boldsymbol{\lambda}) = \frac{1}{n} \left( \boldsymbol{\lambda}^T \boldsymbol{f}(\boldsymbol{X}_n, \boldsymbol{Y}_n) - \log \sum_{\boldsymbol{y}_n \in \mathcal{Y}^{n+1}} \exp\left( \boldsymbol{\lambda}^T \boldsymbol{f}(\boldsymbol{X}_n, \boldsymbol{y}_n) \right) \right).$$

Note that, in the context of finite Conditional Markov Chains, $\mathcal{L}_n(\boldsymbol{\lambda})$ is the exact likelihood of $\boldsymbol{Y}_n$ conditional on $\boldsymbol{X}_n$. The Maximum Likelihood estimate of $\boldsymbol{\lambda}_*$ is given by

$$\hat{\boldsymbol{\lambda}}_n := \arg\max_{\boldsymbol{\lambda} \in \boldsymbol{\Theta}} \mathcal{L}_n(\boldsymbol{\lambda}).$$

If $\mathcal{L}_n(\boldsymbol{\lambda})$ is strictly concave, then the $\arg\max$ is unique and can be found using gradient-based search (see [14]). It is easy to see that $\mathcal{L}_n(\boldsymbol{\lambda})$ is strictly concave if and only if $|\mathcal{Y}| > 1$, and there exists a sequence $\boldsymbol{y}_n$ such that at least one component of $\boldsymbol{f}(\boldsymbol{X}_n, \boldsymbol{y}_n)$ is non-zero. In the following, we study strong consistency of the Maximum Likelihood estimates, a property which is of central importance in large sample theory (see [15]). As we will see, a key problem is the identifiability and uniqueness of the model weights.

## 5.1 Asymptotic properties of the likelihood function

In addition to the conditions (A1)-(A4) stated earlier, we will make the following assumptions:

(A5) The feature functions have finite second moments: $\mathbb{E}_{\boldsymbol{\lambda}_*}[|f(X_t, Y_{t-1}, Y_t)|^2] < \infty$.

(A6) The parameter space $\boldsymbol{\Theta}$ is compact.

The next theorem establishes asymptotic properties of the likelihood function $\mathcal{L}_n(\boldsymbol{\lambda})$.

**Theorem 5.** *Suppose that conditions* (A1)-(A6) *are satisfied. Then the following holds true:*

(i) *There exists a function $\mathcal{L}(\boldsymbol{\lambda})$ such that $\lim_{n\to\infty} \mathcal{L}_n(\boldsymbol{\lambda}) = \mathcal{L}(\boldsymbol{\lambda})$ $\mathbb{P}_{\boldsymbol{\lambda}_*}$-almost surely for every $\boldsymbol{\lambda} \in \boldsymbol{\Theta}$. Moreover, the convergence of $\mathcal{L}_n(\boldsymbol{\lambda})$ to $\mathcal{L}(\boldsymbol{\lambda})$ is uniform on $\boldsymbol{\Theta}$, that is, $\lim_{n\to\infty} \sup_{\boldsymbol{\lambda}\in\boldsymbol{\Theta}} |\mathcal{L}_n(\boldsymbol{\lambda}) - \mathcal{L}(\boldsymbol{\lambda})| = 0$ $\mathbb{P}_{\boldsymbol{\lambda}_*}$-almost surely.*

(ii) *The gradients satisfy $\lim_{n\to\infty} \nabla\mathcal{L}_n(\boldsymbol{\lambda}) = \mathbb{E}_{\boldsymbol{\lambda}_*}[f(X_t, Y_{t-1}, Y_t)] - \mathbb{E}_{\boldsymbol{\lambda}}[f(X_t, Y_{t-1}, Y_t)]$ $\mathbb{P}_{\boldsymbol{\lambda}_*}$-almost surely for every $\boldsymbol{\lambda} \in \boldsymbol{\Theta}$.*

(iii) *If the limit of the Hessian $\nabla^2\mathcal{L}_n(\boldsymbol{\lambda})$ is finite and non-singular, then the function $\mathcal{L}(\boldsymbol{\lambda})$ is strictly concave on $\boldsymbol{\Theta}$. As a consequence, the Maximum Likelihood estimates are strongly consistent:*

$$\lim_{n\to\infty} \hat{\boldsymbol{\lambda}}_n = \boldsymbol{\lambda}_* \qquad \mathbb{P}_{\boldsymbol{\lambda}_*}\text{-almost surely.}$$

*Proof.* The statements are obtained analogously to Lemma 4-6 and Theorem 4 in [1], using the auxiliary results for Conditional Markov Chains with unbounded feature functions established in Theorem 1, Theorem 2, and Theorem 4. $\qquad\square$

Next, we study the asymptotic behaviour of the Hessian $\nabla^2\mathcal{L}_n(\boldsymbol{\lambda})$. In order to compute the derviatives, introduce the vectors $\boldsymbol{\lambda}_1, \ldots, \boldsymbol{\lambda}_n$ with $\boldsymbol{\lambda}_t = \boldsymbol{\lambda}$ for $t = 1, \ldots, n$. This allows us to write $\boldsymbol{\lambda}^T f(\boldsymbol{X}_n, \boldsymbol{Y}_n) = \sum_{t=1}^n \boldsymbol{\lambda}_t^T f(X_t, Y_{t-1}, Y_t)$. Now, regard the argument $\boldsymbol{\lambda}$ of the likelihood function as a stacked vector $(\boldsymbol{\lambda}_1, \ldots, \boldsymbol{\lambda}_n)$. Same as in [1], this gives us the expressions

$$\frac{\partial^2}{\partial\boldsymbol{\lambda}_t \partial\boldsymbol{\lambda}_{t+k}^T}\mathcal{L}_n(\boldsymbol{\lambda}) = \frac{1}{n}\mathrm{Cov}_{\boldsymbol{\lambda}}^{(0:n)}\big[f(X_t, Y_{t-1}, Y_t), f(X_{t+k}, Y_{t+k-1}, Y_{t+k})^T \mid \boldsymbol{X}\big]$$

where $\mathrm{Cov}_{\boldsymbol{\lambda}}^{(0:n)}$ is the covariance with respect to the measure $\mathbb{P}_{\boldsymbol{\lambda}}^{(0:n)}$ introduced before Theorem 3. Using these expressions, the Hessian of $\mathcal{L}_n(\boldsymbol{\lambda})$ can be written as

$$\nabla^2\mathcal{L}_n(\boldsymbol{\lambda}) = -\Big(\sum_{t=1}^n \frac{\partial^2}{\partial\boldsymbol{\lambda}_t \partial\boldsymbol{\lambda}_t^T}\mathcal{L}_n(\boldsymbol{\lambda}) + 2\sum_{k=1}^{n-1}\sum_{t=1}^{n-k} \frac{\partial^2}{\partial\boldsymbol{\lambda}_t \partial\boldsymbol{\lambda}_{t+k}^T}\mathcal{L}_n(\boldsymbol{\lambda})\Big).$$

The following theorem establishes an expression for the limit of $\nabla^2\mathcal{L}_n(\boldsymbol{\lambda})$. It differs from the expression given in Lemma 7 of [1], which is incorrect.

**Theorem 6.** *Suppose that conditions* (A1) - (A5) *hold. Then*

$$\lim_{n\to\infty} \nabla^2\mathcal{L}_n(\boldsymbol{\lambda}) = -\Big(\gamma_{\boldsymbol{\lambda}}(0) + 2\sum_{k=1}^{\infty}\gamma_{\boldsymbol{\lambda}}(k)\Big) \qquad \mathbb{P}_{\boldsymbol{\lambda}_*}\text{-almost surely}$$

*where $\gamma_{\boldsymbol{\lambda}}(k) = \mathbb{E}[\mathrm{Cov}_{\boldsymbol{\lambda}}(f(X_t, Y_{t-1}, Y_t), f(X_{t+k}, Y_{t+k-1}, Y_{t+k}) \mid \boldsymbol{X})]$ is the expectation of the conditional covariance (with respect to $\mathbb{P}_{\boldsymbol{\lambda}}$) between $f(X_t, Y_{t-1}, Y_t)$ and $f(X_{t+k}, Y_{t+k-1}, Y_{t+k})$ given $\boldsymbol{X}$. In particular, the limit of $\nabla^2\mathcal{L}_n(\boldsymbol{\lambda})$ is finite.*

*Proof.* The key step is to show the existence of a positive measurable function $U_{\boldsymbol{\lambda}}(k, \boldsymbol{x})$ such that

$$\lim_{n\to\infty}\sum_{k=1}^{n-1}\sum_{t=1}^{n-k}\Big|\frac{\partial^2}{\partial\boldsymbol{\lambda}_t \partial\boldsymbol{\lambda}_{t+k}^T}\mathcal{L}_n(\boldsymbol{\lambda})\Big| \leq \lim_{n\to\infty}\sum_{k=1}^{n-1}\mathbb{E}[U_{\boldsymbol{\lambda}}(k, \boldsymbol{X})]$$

with the limit on the right hand side being finite. Then the rest of the proof is straight-forward: Theorem 4 shows that, for fixed $k \geq 0$,

$$\lim_{n \to \infty} \sum_{t=1}^{n-k} \frac{\partial^2}{\partial \boldsymbol{\lambda}_t \partial \boldsymbol{\lambda}_{t+k}^T} \mathcal{L}_n(\boldsymbol{\lambda}) \;\; = \;\; \gamma_{\boldsymbol{\lambda}}(k) \qquad \mathbb{P}_{\boldsymbol{\lambda}_*}\text{-almost surely.}$$

Hence, in order to establish the theorem, it suffices to show that

$$\lim_{n \to \infty} \sum_{k=1}^{n-1} \left| \gamma_{\boldsymbol{\lambda}}(k) - \sum_{t=1}^{n-k} \frac{\partial^2}{\partial \boldsymbol{\lambda}_t \partial \boldsymbol{\lambda}_{t+k}^T} \mathcal{L}_n(\boldsymbol{\lambda}) \right| \;\; \leq \;\; \epsilon$$

for all $\epsilon > 0$. Now let $\epsilon > 0$ be fixed. According to Theorem 2 we have $\sum_{k=1}^{\infty} |\gamma_{\boldsymbol{\lambda}}(k)| < \infty$. Hence we can find a finite $N$ such that

$$\lim_{n \to \infty} \sum_{k=N}^{n-1} |\gamma_{\boldsymbol{\lambda}}(k)| + \lim_{n \to \infty} \sum_{k=N}^{n-1} \mathbb{E}[U_{\boldsymbol{\lambda}}(k, \boldsymbol{X})] \;\; \leq \;\; \epsilon.$$

On the other hand, Theorem 4 shows that

$$\lim_{n \to \infty} \sum_{k=1}^{N-1} \left| \gamma_{\boldsymbol{\lambda}}(k) - \sum_{t=1}^{n-k} \frac{\partial^2}{\partial \boldsymbol{\lambda}_t \partial \boldsymbol{\lambda}_{t+k}^T} \mathcal{L}_n(\boldsymbol{\lambda}) \right| \;\; = \;\; 0.$$

For details on how to construct $U_{\boldsymbol{\lambda}}(k, \boldsymbol{x})$, see the extended version of this paper. $\qquad \square$

## 5.2 Model identifiability

Let us conclude the analysis by investigating conditions under which the limit of the Hessian $\nabla^2 \mathcal{L}_n(\boldsymbol{\lambda})$ is non-singular. Note that $\nabla^2 \mathcal{L}_n(\boldsymbol{\lambda})$ is negative definite for every $n$, so also the limit is negative definite, but not necessarily strictly negative definite. Using the result in Theorem 6, we can establish the following statement:

**Corollary 1.** *Suppose that assumptions* (A1)-(A5) *hold true. Then the following conditions are necessary for the limit of* $\nabla^2 \mathcal{L}_n(\boldsymbol{\lambda})$ *to be non-singular:*

(i) *For each feature function* $f(x, i, j)$, *there exists a set* $A \subset \mathcal{X}$ *with* $\mathbb{P}(X_t \in A) > 0$ *such that, for every* $x \in A$, *we can find* $i, j, k, l \in \mathcal{Y}$ *with* $f(x, i, j) \neq f(x, k, l)$.

(ii) *There does not exist a non-zero vector* $\boldsymbol{\lambda}$ *and a subset* $A \subset \mathcal{X}$ *with* $\mathbb{P}(X_t \in A) = 1$ *such that* $\boldsymbol{\lambda}^T \boldsymbol{f}(x, i, j)$ *is constant for all* $x \in \mathcal{X}$ *and* $i, j \in \mathcal{Y}$.

Condition $(i)$ essentially says: features $f(x, i, j)$ must not be constant in $i$ and $j$. Condition $(ii)$ says that features must not be expressible as linear combinations of each other. Clearly, features violating condition $(i)$ can be assigned arbitrary model weights without any effect on the conditional distributions. If condition $(ii)$ is violated, then there are infinitely many ways for parameterizing the same model. In practice, some authors have found positive effects of including redundant features (see, e.g., [16]), however, usually in the context of a learning objective with an additional penalizer.

## 6 Conclusions

We have established ergodicity and various mixing properties of Conditional Markov Chains with unbounded feature functions. The main insight is that similar results to the setting with bounded feature functions can be obtained, however, under additional assumptions on the distribution of the observations. In particular, the proof of Theorem 2 has shown that the sequence of observations needs to satisfy conditions under which Hoeffding-type concentration inequalities can be obtained. The proof of Theorem 3 has reveiled an interesting interplay between mixing rates, feature functions, and the tail behaviour of the distribution of observations. By applying the mixing properties to the empirical likelihood functions we have obtained necessary conditions for the Maximum Likelihood estimates to be strongly consistent. We see a couple of interesting problems for future research: establishing Central Limit Theorems for Conditional Markov Chains; deriving bounds for the asymptotic variance of Maximum Likelihood estimates; constructing tests for the significance of features; generalizing the estimation theory to an infinite number of features; finally, finding sufficient conditions for the model identifiability.

# References

[1] Sinn, M. & Poupart, P. (2011) Asymptotic theory for linear-chain conditional random fields. In *Proc. of the 14th International Conference on Artificial Intelligence and Statistics (AISTATS)*.

[2] Lafferty, J., McCallum, A. & Pereira, F. (2001) Conditional random fields: Probabilistic models for segmenting and labeling sequence data. In *Proc. of the 18th IEEE International Conference on Machine Learning (ICML)*.

[3] Sutton, C. & McCallum, A. (2006) An introduction to conditional random fields for relational learning. In: Getoor, L. & Taskar, B. (editors), *Introduction to Statistical Relational Learning*. Cambridge, MA: MIT Press.

[4] Hofmann, T., Schölkopf, B. & Smola, A.J. (2008) Kernel methods in machine learning. *The Annals of Statstics*, Vol. 36, No. 3, 1171-1220.

[5] Xiang, R. & Neville, J. (2011) Relational learning with one network: an asymptotic analysis. In *Proc. of the 14th International Conference on Artificial Intelligence and Statistics (AISTATS)*.

[6] Seneta, E. (2006) *Non-Negative Matrices and Markov Chains. Revised Edition.* New York, NY: Springer.

[7] Wainwright, M.J. & Jordan, M.I. (2008) Graphical models: exponential families, and variational inference. *Foundations and Trends $^®$ in Machine Learning*, Vol. 1, Nos. 1-2, 1-305.

[8] Cornfeld, I.P., Fomin, S.V. & Sinai, Y.G. (1982) *Ergodic Theory.* Berlin, Germany: Springer.

[9] Orey, S. (1991) Markov chains with stochastically stationary transition probabilities. *The Annals of Probability*, Vol. 19, No. 3, 907-928.

[10] Hernández-Lerma, O. & Lasserre, J.B. (2003) *Markov Chains and Invariant Probabilities.* Basel, Switzerland: Birkhäuser.

[11] Foguel, S.R. (1969) *The Ergodic Theory of Markov Processes.* Princeton, NJ: Van Nostrand.

[12] Samson, P.-M. (2000) Concentration of measure inequalities for Markov chains and $\Phi$-mixing processes. *The Annals of Probability*, Vol. 28, No. 1, 416-461.

[13] Kontorovich, L. & Ramanan, K. (2008) Concentration inequalities for dependent random variables via the martingale method. *The Annals of Probability*, Vol. 36, No. 6, 2126-2158.

[14] Sha, F. & Pereira, F. (2003) Shallow parsing with conditional random fields. In *Proc. of the Human Language Technology Conference of the North American Chapter of the Association for Computational Linguistics (HLT-NAACL)*.

[15] Lehmann, E.L. (1999) *Elements of Large-Sample Theory.* New York, NY: Springer.

[16] Hoefel, G. & Elkan, C. (2008) Learning a two-stage SVM/CRF sequence classifier. In *Proc. of the 17th ACM International Conference on Information and Knowledge Management (CIKM)*.
